# Phonetic Speaker Recognition with Support Vector Machines

**W. M. Campbell, J. P. Campbell, D. A. Reynolds, D. A. Jones, and T. R. Leek**
MIT Lincoln Laboratory
Lexington, MA 02420
*wcampbell,jpc,dar,daj,tleek@ll.mit.edu*

## Abstract

A recent area of significant progress in speaker recognition is the use of high level features—idiolect, phonetic relations, prosody, discourse structure, etc. A speaker not only has a distinctive acoustic sound but uses language in a characteristic manner. Large corpora of speech data available in recent years allow experimentation with long term statistics of phone patterns, word patterns, etc. of an individual. We propose the use of support vector machines and term frequency analysis of phone sequences to model a given speaker. To this end, we explore techniques for text categorization applied to the problem. We derive a new kernel based upon a linearization of likelihood ratio scoring. We introduce a new phone-based SVM speaker recognition approach that halves the error rate of conventional phone-based approaches.

## 1 Introduction

We consider the problem of text-independent speaker verification. That is, given a claim of identity and a voice sample (whose text content is *a priori* unknown), determine if the claim is correct or incorrect. Traditional speaker recognition systems use features based upon the spectral content (e.g., cepstral coefficients) of the speech. Implicitly, these systems model the vocal tract and its associated dynamics over a short time period. These approaches have been quite successful, see [1, 2].

Traditional systems have several drawbacks. First, robustness is an issue because channel effects can dramatically change the measured acoustics of a particular individual. For instance, a system relying only on acoustics might have difficulty confirming that an individual speaking on a land-line telephone is the the same as an individual speaking on a cell phone [3]. Second, traditional systems also rely upon seemingly different methods than human listeners [4]. Human listeners are aware of prosody, word choice, pronunciation, accent, and other speech habits (laughs, etc.) when recognizing speakers. Potentially because of this use of higher level cues, human listeners are less affected by variation in channel and noise types than automatic algorithms.

An exciting area of recent development pioneered by Doddington [5] is the use of "high level" features for speaker recognition. In Doddington's idiolect work, word $N$-grams from conversations were used to characterize a particular speaker. More recent systems have used a variety of approaches involving phone sequences [6], pronunciation modeling [7],

and prosody [8]. For this paper, we concentrate on the use of phone sequences [6]. The processing for this type of system uses acoustic information to obtain sequences of phones for a given conversation and then discards the acoustic waveform. Thus, processing is done at the level of terms (symbols) consisting of, for example, phones or phone $N$-grams.

This paper is organized as follows. In Section 2, we discuss the NIST extended data speaker recognition task. In Section 3.1, we present a method for obtaining a phone stream. Section 3.2 shows the structure of the SVM phonetic speaker recognition system. Section 4 discusses how we construct a kernel for speaker recognition using term weighting techniques for document classification. We derive a new kernel based upon a linearization of a likelihood ratio. Finally, Section 5 shows the applications of our methods and illustrates the dramatic improvement in performance possible over standard phone-based $N$-gram speaker recognition methods.

## 2   The NIST extended data task

Experiments for the phone-based speaker recognition experiments were performed based upon the NIST 2003 extended data task [9]. The corpus used was a combination of phases II and III of the Switchboard-2 corpora [10].

Each potential training utterance in the NIST extended data task consisted of a conversation side that was nominally of length $5$ minutes recorded over a land-line telephone. Each conversation side consisted of a speaker having a conversation on a topic selected by an automatic operator; conversations were typically between unfamiliar individuals.

For training and testing, a jacknife approach was used to increase the number of tests. The data was divided into $10$ splits. For training, a given split contains speakers to be recognized (target speakers) and impostor speakers; the remaining splits could be used to construct models describing the statistics of the general population—a "background" model. For example, when conducting tests on split 1, splits 2-10 could be used to construct a background.

Training a speaker model was performed by using statistics from $1$, $2$, $4$, $8$, or $16$ conversation sides. This simulated a situation where the system could use longer term statistics and become "familiar" with the individual; this longer term training allows one to explore techniques which might mimic more what human listeners perceive about an individual's speech habits. A large number of speakers and tests were available; for instance, for 8 conversation training, 739 distinct target speakers were used and $11,171$ true trials and $17,736$ false trials were performed. For additional information on the training/testing structure we refer to the NIST extended data task description [9].

## 3   Speaker Recognition with Phone Sequences

### 3.1   Phone Sequence Extraction

Phone sequence extraction for the speaker recognition process is performed using the phone recognition system (PPRLM) designed by Zissman [11] for language identification. PPRLM uses a mel-frequency cepstral coefficient front end with delta coefficients. Each phone is modeled in a gender-dependent context-independent (monophone) manner using a three-state hidden Markov model (HMM). Phone recognition is performed with a Viterbi search using a fully connected null-grammar network of monophones; note that no explicit language model is used in the decoding process.

The phone recognition system encompassed multiple languages—English (EG), German (GE), Japanese (JA), Mandarin (MA), and Spanish (SP). In earlier phonetic speaker recog-

nition work [6], it was found that these multiple streams were useful for improving accuracy. The phone recognizers were trained using the OGI multilanguage corpus which had been hand labeled by native speakers.

After a "raw" phone stream was obtained from PPRLM, additional processing was performed to increase robustness. First, speech activity detection marks were used to eliminate phone segments where no speech was present. Second, silence labels of duration greater than $0.5$ seconds were replaced by "end start" pairs. The idea in this case is to capture some of the ways in which a speaker interacts with others—does the speaker pause frequently, etc. Third, extraneous silence was removed at the beginning and end of the resulting segments. Finally, all phones with short duration were removed (less than 3 frames).

### 3.2 Phonetic SVM System

Our system for speaker recognition using phone sequences is shown in Figure 1. The scenario for its usage is as follows. An individual makes a claim of identity. The system then retrieves the SVM models of the claimed identity for each of the languages in the system. Speech from the individual is then collected (a test utterance). A phone sequence is derived using each of the language phone recognizers and then post-processing is performed on the sequence as discussed in Section 3.1. After this step, the phone sequence is vectorized by computing frequencies of $N$-grams—this process will be discussed in Section 4. We call this term calculation since we compute term types (unigram, bigram, etc.), term probabilities and weightings in this step [12]. This vector is then introduced into a SVM using the speaker's model in the appropriate language and a score per language is produced. Note that we do not threshold the output of the SVM. These scores are then fused using a linear weighting to produce a final score for the test utterance. The final score is compared to a threshold and a reject or accept decision is made based upon whether the score was below or above the threshold, respectively.

An interesting aspect of the system in Figure 1 is that it uses multiple streams of phones in different languages. There are several reasons for this strategy. First, the system can be used without modification for speakers in multiple languages. Second, although not obvious, from experimentation we show that phone streams different from the language being spoken provide complimentary information for speaker recognition. That is, accuracy improves with these additional systems. A third point is that the system may also work in other languages not represented in the phone streams. It is known that in the case of language identification, language characterization can be performed even if a phone recognizer is not available in that particular language [11].

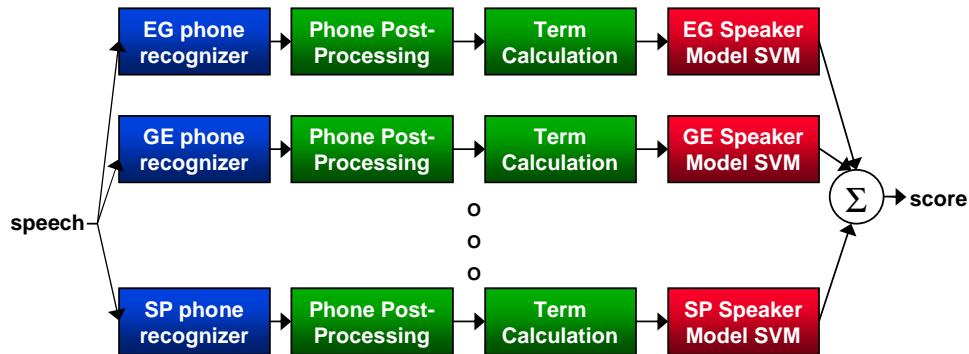

Figure 1: Phonetic speaker recognition using support vector machines

Training for the system in Figure 1 is based upon the structure of the NIST extended data corpus (see Section 2). We treat each conversation side in the corpus as a "document." From each of these conversation sides we derive a single (sparse) vector of weighted probabilities. To train a model for a given speaker, we use a one-versus-all strategy. The speaker's conversations are trained to a SVM target value of $+1$. The conversations sides not in the current split (see Section 2) are used as a background. That is, all conversation sides not in the current split are used as the class for SVM target value of $-1$. Note that this strategy ensures that speakers that are used as impostors are "unseen" in the training data.

## 4   Kernel Construction

Possibly the most important aspect of the process of phonetic speaker recognition is the selection of the kernel for the SVM. Of particular interest is a kernel which will preserve the identity cues a particular individual might present in their phone sequence. We describe two steps for kernel construction.

Our first step of kernel construction is the selection of probabilities to describe the phone stream. We follow the work of [5, 6]. The basic idea is to use a "bag of $N$-grams" approach. For a phone sequence, we produce $N$-grams by the standard transformation of the stream; e.g., for bigrams (2-grams) the sequence of phones, $t_1$, $t_2$, ..., $t_n$, is transformed to the sequence of bigrams of phones $t_1\_t_2$, $t_2\_t_3$, ..., $t_{n-1}\_t_n$. We then find probabilities of $N$-grams with $N$ fixed. That is, suppose we are considering unigrams and bigrams of phones, and the unique unigrams and bigrams are designated $d_1$, ..., $d_M$ and $d_1\_d_1$, ... $d_M\_d_M$, respectively; then we calculate probabilities and joint probabilities

$$p(d_i) = \frac{\#(t_k = d_i)}{\sum_k \#(t_k = d_k)}$$
$$p(d_i\_d_j) = \frac{\#(t_k\_t_l = d_i\_d_j)}{\sum_{k,l} \#(t_i\_t_j = d_k\_d_l)} \tag{1}$$

where $\#(t_k = d_i)$ indicates the number of phones in the conversation side equal to $d_i$, and an analogous definition is used for bigrams. These probabilities then become entries in a vector $\mathbf{v}$ describing the conversation side

$$\mathbf{v} = [p(d_1) \quad \ldots \quad p(d_M) \quad p(d_1\_d_1) \quad \ldots \quad p(d_M\_d_M)]^t. \tag{2}$$

In general, the vector $\mathbf{v}$ will be sparse since the conversation side will not contain all potential unigrams, bigrams, etc.

A second step of kernel construction is the selection of the "document component" of term weighting for the entries of the vector $\mathbf{v}$ in (2) and the normalization of the resulting vector. By term weighting we mean that for each entry, $v_i$, of the vector $\mathbf{v}$, we multiply by a "collection" (or background) component, $w_i$, for that entry. We tried two distinct approaches for term weighting.

**TFIDF weighting.** The first is based upon the standard TFIDF approach [12, 13]. From the background section of the corpus we compute the frequency of a particular $N$-gram using conversation sides as the item analogous to a document. I.e., if we let $DF(t_i)$ be the number of conversation sides where a particular $N$-gram, $t_i$, is observed, then our resulting term-weighted vector has entries

$$v_i \log \left( \frac{\# \text{ of conversation sides in background}}{DF(t_i)} \right). \tag{3}$$

We follow the weighting in (3) by a normalization of the vector to unit length $\mathbf{x} \mapsto \mathbf{x}/\|\mathbf{x}\|_2$.

**Log-likelihood ratio weighting.** An alternate method of term weighting may be derived using the following strategy. Suppose that we have two conversation sides from speakers, $\text{spk}_1$ and $\text{spk}_2$. Further suppose that the sequence of $N$-grams (for fixed $N$) in each conversation side is $t_1, t_2, ..., t_n$ and $u_1, u_2, ..., u_m$ respectively. We denote the unique set of $N$-grams as $d_1, ..., d_M$. We can build a "model" based upon the conversation sides for each speaker consisting of the probability of $N$-grams, $p(d_i|\text{spk}_j)$. We then compute the likelihood ratio of the first conversation side as is standard in verification problems [1]; a linearization of the likelihood ratio computation will serve as the kernel. Proceeding,

$$\frac{p(t_1, t_2, \ldots, t_n|\text{spk}_2)}{p(t_1, \ldots, t_n|\text{background})} = \prod_{i=1}^{n} \frac{p(t_i|\text{spk}_2)}{p(t_i|\text{background})} \tag{4}$$

where we have made the assumption that the probabilities are independent. We then consider the $\log$ of the likelihood ratio normalized by the number of observations,

$$\begin{aligned}
\text{score} &= \frac{1}{n} \sum_{i=1}^{n} \log\left( \frac{p(t_i|\text{spk}_2)}{p(t_i|\text{background})} \right) \\
&= \sum_{j=1}^{M} \frac{\#(t_i = d_j)}{n} \log\left( \frac{p(d_j|\text{spk}_2)}{p(d_j|\text{background})} \right) \\
&= \sum_{j=1}^{M} p(d_j|\text{spk}_1) \log\left( \frac{p(d_j|\text{spk}_2)}{p(d_j|\text{background})} \right).
\end{aligned} \tag{5}$$

If we now "linearize" the $\log$ function in (5) by using $\log(x) \approx x - 1$, we get

$$\begin{aligned}
\text{score} &\approx \sum_{j=1}^{M} p(d_j|\text{spk}_1) \frac{p(d_j|\text{spk}_2)}{p(d_j|\text{background})} - \sum_{j=1}^{M} p(d_j|\text{spk}_1) \\
&= \sum_{j=1}^{M} p(d_j|\text{spk}_1) \frac{p(d_j|\text{spk}_2)}{p(d_j|\text{background})} - 1 \\
&= \sum_{j=1}^{M} \frac{p(d_j|\text{spk}_1)}{\sqrt{p(d_j|\text{background})}} \frac{p(d_j|\text{spk}_2)}{\sqrt{p(d_j|\text{background})}} - 1
\end{aligned} \tag{6}$$

Thus, (6) suggests we use a term weighting given by $1/\sqrt{p(d_j|\text{background})}$. Note that the strategy used for constructing a kernel is part of a general process of finding kernels based upon training on one instance and testing upon another instance [2].

## 5 Experiments

Experiments were performed using the NIST extended data task "v1" lists (which encompass the entire Switchboard 2 phase II and III corpora). Tests were performed for 1, 2, 4, 8, and 16 training conversations. Scoring was performed using the SVM system shown in Figure 1. Five language phone recognizers were used—English (EG), German (GE), Japanese (JA), Mandarin (MA), and Spanish (SP). The resulting phone sequences were vectorized as unigram and bigram probabilities (2). Both the standard TFIDF term weighting (3) and the log-likelihood ratio (TFLLR) term weighting (6) methods were used. We note that when a term did not appear in the background, it was ignored in training and scoring. A linear kernel was used $\mathbf{x} \cdot \mathbf{y} + 1$ to compare the vectors of term weights. Training was performed using the SVMTorch package [14] with $c = 1$. Comparisons of performance for different strategies were typically done with $8$ conversation training and English phone streams since these were representative of overall performance.

Table 1: Comparison of different term weighting strategies, English only scores, 8 conversation training

| Term Weighting Method | EER |
|:---:|:---:|
| TFIDF | 7.4% |
| TFLLR | 5.2% |

Results were compared via equal error rates (EERs)—the error at the threshold which produces equal miss and false alarm probabilities, $P_{\mathrm{miss}} = P_{\mathrm{fa}}$. Table 1 shows the results for two different weightings, TFIDF (3) and TFLLR (6), using English phones only and 8 training conversations. The table illustrates that the new TFLLR weighting method is more effective. This may be due to the fact the IDF is too "smooth"; e.g., for unigrams, the IDF is approximately 1 since a unigram almost always appears in a given 5 minute conversation. Also, alternate methods of calculating the TF component of TFIDF have not been explored and may yield gains compared to our formulation.

We next considered the effect on performance of the language of the phone stream for the 8 conversation training case. Figure 2 shows a DET plot (a ROC plot with a special scale [15]) with results corresponding to the 5 language phone streams. The best performing system in the figure is an equal fusion of all scores from the SVM outputs for each language and has an EER of 3.5%; other fusion weightings were not explored in detail. Note that the best performing language is English, as expected. Note, though, as we indicated in Section 3.1 that other languages do provide significant speaker recognition information.

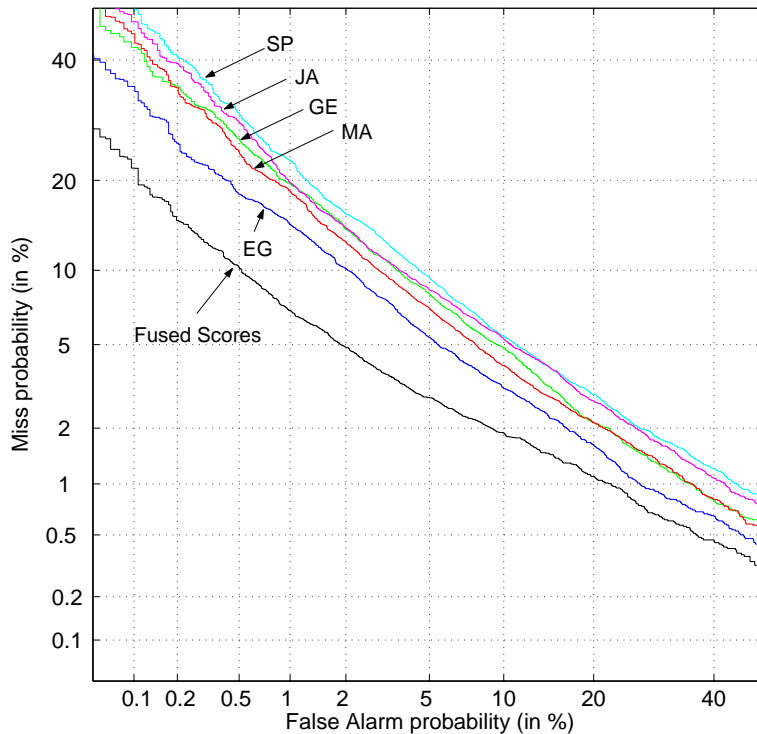

Figure 2: DET plot for the 8 conversation training case with varying languages and TFLLR weighting. The plot shows in order of increasing EER—fused scores, EG, MA, GE, JA, SP

Table 2: Comparison of equal error rates (EERs) for different conversation training lengths using the TFLLR phonetic SVM and the standard log likelihood ratio (LLR) method

| # Training Conversations | SVM EER | LLR EER | SVM EER Reduction |
|---|---|---|---|
| 1 | 13.4% | 21.8% | 38% |
| 2 | 8.6% | 14.9% | 42% |
| 4 | 5.3% | 10.3% | 49% |
| 8 | 3.5% | 8.8% | 60% |
| 16 | 2.5% | 8.3% | 70% |

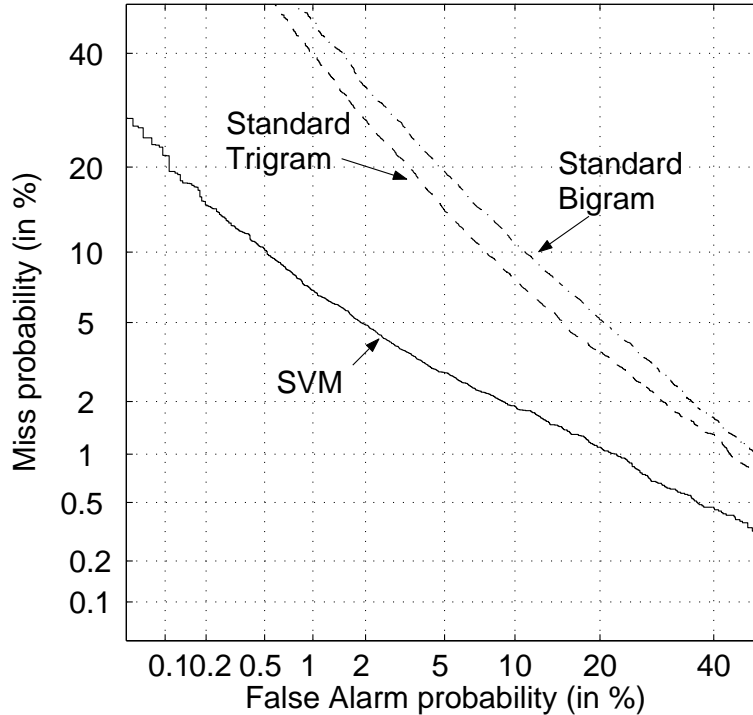

Figure 3: DET plot for 8 conversation training showing a comparison of the SVM approach (solid line) to the standard log likelihood ratio approach using bigrams (dash-dot line) and the standard log likelihood ratio approach using trigrams (dashed line)

Table 2 shows the effect of different training conversation lengths on the EER. As expected, more training data leads to lower error rates. We also see that even for 1 training conversation, the SVM system provides significant speaker characterization ability. Figure 3 shows DET plots comparing the performance of the standard log likelihood ratio method [6] to our new SVM method using the TFLLR weighting. We show log likelihood results based on both bigrams and trigrams; in addition, a slightly more complex model involving discounting of probabilities is used. One can see the dramatic reduction in error, especially apparent for low false alarm probabilities. The EERs of the standard system are $8.8\%$ (trigrams, see Table 2) and $10.4\%$ (bigrams), whereas our new SVM system produces an EER of $3.5\%$; thus, we have reduced the error rate by $60\%$.

# 6 Conclusions and future work

An exciting new application of SVMs to speaker recognition was shown. By computing frequencies of phones in conversations, speaker characterization was performed. A new kernel was introduced based on the standard method of log likelihood ratio scoring. The resulting SVM method reduced error rates dramatically over standard techniques.

### Acknowledgements

This work was sponsored by the United States Government Technical Support Working Group under Air Force Contract F19628-00-C-0002. Opinions, interpretations, conclusions, and recommendations are those of the authors and are not necessarily endorsed by the United States Government.

# References

[1] Douglas A. Reynolds, T. F. Quatieri, and R. Dunn, "Speaker verification using adapted Gaussian mixture models," *Digital Signal Processing*, vol. 10, no. 1-3, pp. 19–41, 2000.

[2] W. M. Campbell, "Generalized linear discriminant sequence kernels for speaker recognition," in *Proceedings of the International Conference on Acoustics Speech and Signal Processing*, 2002, pp. 161–164.

[3] T. F. Quatieri, D. A. Reynolds, and G. C. O'Leary, "Estimation of handset nonlinearity with application to speaker recognition," *IEEE Trans. Speech and Audio Processing*, vol. 8, no. 5, pp. 567–584, 2000.

[4] Astrid Schmidt-Nielsen and Thomas H. Crystal, "Speaker verification by human listeners: Experiments comparing human and machine performance using the NIST 1998 speaker evaluation data," *Digital Signal Processing*, vol. 10, pp. 249–266, 2000.

[5] G. Doddington, "Speaker recognition based on idiolectal differences between speakers," in *Proceedings of Eurospeech*, 2001, pp. 2521–2524.

[6] Walter D. Andrews, Mary A. Kohler, Joseph P. Campbell, John J. Godfrey, and Jaime Hernández-Cordero, "Gender-dependent phonetic refraction for speaker recognition," in *Proceedings of the International Conference on Acoustics Speech and Signal Processing*, 2002, pp. I149–I153.

[7] David Klusáček, Jirí Navarátil, D. A. Reynolds, and J. P. Campbell, "Conditional pronunciation modeling in speaker detection," in *Proceedings of the International Conference on Acoustics Speech and Signal Processing*, 2003, pp. IV–804–IV–807.

[8] Andre Adami, Radu Mihaescu, Douglas A. Reynolds, and John J. Godfrey, "Modeling prosodic dynamics for speaker recognition," in *Proceedings of the International Conference on Acoustics Speech and Signal Processing*, 2003, pp. IV–788–IV–791.

[9] M. Przybocki and A. Martin, "The NIST year 2003 speaker recognition evaluation plan," http://www.nist.gov/speech/tests/spk/2003/index.htm, 2003.

[10] Linguistic Data Consortium, "Switchboard-2 corpora," http://www.ldc.upenn.edu.

[11] M. Zissman, "Comparison of four approaches to automatic language identification of telephone speech," *IEEE Trans. Speech and Audio Processing*, vol. 4, no. 1, pp. 31–44, 1996.

[12] Thorsten Joachims, *Learning to Classify Text Using Support Vector Machines*, Kluwer Academic Publishers, 2002.

[13] G. Salton and C. Buckley, "Term weighting approaches in automatic text retrieval," *Information Processing and Management*, vol. 24, no. 5, pp. 513–523, 1988.

[14] Ronan Collobert and Samy Bengio, "SVMTorch: Support vector machines for large-scale regression problems," *Journal of Machine Learning Research*, vol. 1, pp. 143–160, 2001.

[15] Alvin Martin, G. Doddington, T. Kamm, M. Ordowski, and Marc Przybocki, "The DET curve in assessment of detection task performance," in *Proceedings of Eurospeech*, 1997, pp. 1895–1898.
